# Translating Locative Prepositions

**Paul W. Munro and Mary Tabasko**
Department of Information Science
University of Pittsburgh
Pittsburgh, PA 15260

## ABSTRACT

A network was trained by back propagation to map locative expressions of the form "noun-preposition-noun" to a semantic representation, as in Cosic and Munro (1988). The network's performance was analyzed over several simulations with training sets in both English and German. Translation of prepositions was attempted by presenting a locative expression to a network trained in one language to generate a semantic representation; the semantic representation was then presented to the network trained in the other language to generate the appropriate preposition.

## 1 INTRODUCTION

Connectionist approaches have enjoyed success, relative to competing frameworks, in accounting for context sensitivity and have become an attractive approach to NLP. An architecture (Figure 1) was put forward by Cosic and Munro (1988) to map locative expressions of the form "noun-preposition-noun" to a representation of the spatial relationship between the referents of the two nouns. The features used in the spatial representations were abstracted from Herskovits (1986). The network was trained using the generalized delta rule (Rumelhart, Hinton, and Williams, 1986) on a set of patterns with four components, three syntactic and one semantic. The syntactic components are a pair of nouns separated by a locative preposition [N1-LP-N2], and the semantic component is a representation of the spatial relationship [SR].

The architecture of the network includes two encoder banks, E1 and E2, inspired by Hinton (1986), to force the development of distributed representations of the nouns. This was not done to enhance the performance of the network but rather to facilitate analysis of the network's function, since an important component of Herskovits' theory is the role of nouns as modifiers of the preposition's ideal meaning.

The networks were trained to perform a pattern-completion task. That is, three components from a pattern are selected from the training set and presented to the input layer; either the LP or the SR component is missing. The task is to provide both the LP and SR components at the output. Analysis of a network after the learning phase consists of several tests, such as presenting prepositions with no accompanying nouns, in order to obtain an "ideal meaning" for each preposition, and comparing the noun representations at the encoder banks E1 and E2.

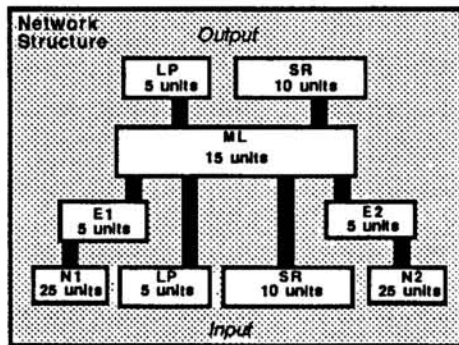

| Noun Units (25) | clouds | lake | campsite | table | book |
|---|---|---|---|---|---|
| | sky | river | school | glass | flowers |
| | plane | road | house | bowl | grass |
| | boat | city | floor | crack | man |
| | water | island | room | chip | fish |

| Spatial Relation Units (10) | | |
|---|---|---|
| N1 over N2 | | N1 within border of N2 |
| N2 over N1 | | N1 touching N2 |
| N1 at edge of N2 | | N1 near N2 |
| N1 embedded in N2 | | N1 far from N2 |
| N2 contains N1 | | N2 supports N1 |

| Preposition Units (5) | | | | |
|---|---|---|---|---|
| in | at | on | under | above |

Figure 1: Network Architecture. Inputs are presented at the lowest layer, either across input banks N1, LP, and N2 or across input banks N1, SR, and N2. The bold lines indicate connectivity from all the units in the lower bank to all the units in the upper bank. The units used to represent the patterns are listed in the table on the right.

## 2 METHODOLOGY

### 2.1 THE TRAINING SETS

3125 (25 x 5 x 25) pattern combinations can be formed with the 25 nouns and five prepositions; of these, 137 meaningful expressions were chosen to constitute an English "training corpus". For each phrase, a set of one to three SR units was chosen to represent the position of the second noun's referent relative to the first noun's. To generate the German corpus, we picked the best German preposition to describe the spatial representation between the nouns. So, each training set consists of the same set of 137 spatial relationships between pairs of nouns. The correspondences between prepositions in the two languages across training sets is given in Table 1.

Table 1:   The number of correspondences between the prepositions used in the English and German training sets.

| ENG GER | IN | AT | ON | UNDER | ABOVE |
|---|---|---|---|---|---|
| IN | 53 | 4 | 0 | 0 | 0 |
| AN | 0 | 9 | 12 | 0 | 0 |
| AUF | 0 | 8 | 20 | 0 | 0 |
| UNTER | 0 | 0 | 0 | 18 | 0 |
| ÜBER | 0 | 0 | 0 | 0 | 13 |

## 2.2 TRANSLATION OF THE PREPOSITIONS

Transforming syntactic expressions to semantic representations and inverting the process in another language is known as the interlingua approach to machine translation. The network described in this paper is particularly well-suited to this approach since it can perform this transformation in either direction (encoding or decoding). Networks trained using expressions from two languages can be attached in sequence to accomplish the translation task. A syntactic triple (N1-LP-N2) from the source language is presented to the network trained in that language. The resulting SR output is then presented with the corresponding nouns in the target language as input to the network trained in the target language, yielding the appropriate preposition in the target language as output. In this procedure, it is assumed that, relative to the prepositions, the nouns are easy to translate; that is, the translation of the nouns is assumed to be much less dependent on context. An example translation of the preposition *on* in the expression "house on lake" is illustrated in Figure 2.

## 3   RESULTS

Eight networks were trained using the two-stroke procedure described above; four using English language inputs and four using German, with two different learning rates in each language, and two different initializations for the random number generator in each case.

Various tests were performed on the trained network in order to determine the ideal meaning of each preposition, the network's classification of the various nouns, and the contextual interaction of the nouns with the prepositions. Also, translation of prepositions from English to German was attempted. The various test modes are described in detail below.

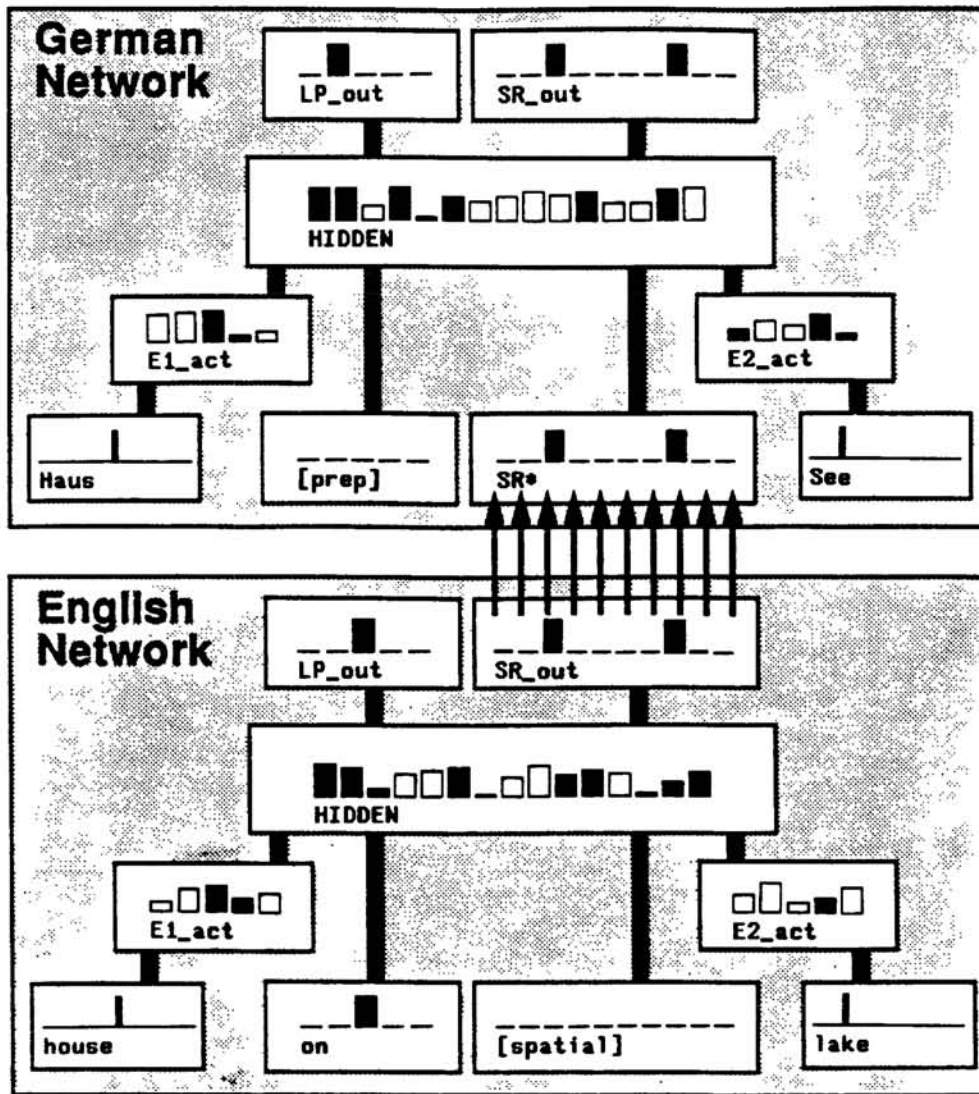

Figure 2: A Schematic View of the Translation Procedure.After training networks in two languages, a preposition can be appropriately translated from one language to the other by performing a decoding task in the source language followed by an encoding task in the target language. The figure shows the resulting activity patterns from the expression "house on lake". The system correctly translates *on* in English to *an* in German. In other contexts, *on* could correspond to the German *auf.*

## 3.1  CONVERGENCE

In each case, the networks converged to states of very low average error (less than 0.5%). However, in no case did a network learn to respond correctly to every phrase in the training set. The performance of each training run was measured by computing the total sum of squared error over the output units across all 137 training patterns. The errors were analyzed into four types:

| | |
|---|---|
| LP-LP errors: | Errors in the LP output units for (N1-LP-N2) input |
| SR-LP errors (encoding): | Errors in the LP output units for (N1-SR-N2) input |
| LP-SR errors (decoding): | Errors in the SR output units for (N1-LP-N2) input |
| SR-SR errors: | Errors in the SR output units for (N1-SR-N2) input |

In assessing the performance of the network after learning, the error measure driving the training (that is, the difference between desired and actual activity levels for every output unit) is inappropriate. In cases such as this, where the output units are being trained to binary values, it is much more informative to compare the relative activity of the output units to the desired pattern and simply count the number of inputs that are "wrong". This approach was used to determine whether each phrase had been processed correctly or incorrectly by the network. Preposition output errors were counted by identifying the most highly activated output unit and checking whether it matched the correct preposition. Since the number of active units in the SR component of each training pattern varies from one to three, a response was registered as incorrect if any of the units that should have been off were more active than any of those that should have been on. These results are reported in Table 2 as total errors out of the 137 in the training corpus.

Table 2: Number of errors for each task in each simulation (out of 137).

|  | LP - LP | SR-LP | LP-SR | SR - SR |
|---|---|---|---|---|
| ENG 1 | 0 | 0 | 3 | 0 |
| ENG 2 | 0 | 0 | 2 | 0 |
| ENG 3 | 0 | 0 | 2 | 0 |
| ENG 4 | 0 | 0 | 2 | 0 |
| ENG AVG | 0.00 | 0.00 | 2.25 | 0.00 |
|  |  |  |  |  |
| GER 1 | 0 | 1 | 2 | 0 |
| GER 2 | 0 | 1 | 3 | 0 |
| GER 3 | 0 | 0 | 2 | 0 |
| GER 4 | 0 | 0 | 4 | 0 |
| GER AVG | 0.00 | 0.50 | 2.75 | 0.00 |

## 3.2 IDEAL MEANINGS OF THE PREPOSITIONS

To find the unmodified spatial representation the net associates with each preposition, the prepositions were presented individually to the net and the resulting spatial responses recorded. This gives a context-free interpretation of each preposition. Figure 3 shows the output activity on the spatial units for one simulation in each language. The results were similar for all simulations within a language, demonstrating that the network finds fairly stable representations for the prepositions. Note that the representations of German *auf*, *an*, and *in* share much of their activation with those of English *on*, *at*, and *in*, although its distribution across the prepositions varies. For example, the preposition *auf* is activated much like English *on*, but without the units indicating the first object at the edge of and near the second. These units are found weakly activated in German *an*, along with the unit indicating coincidence. The ideal meaning of *auf*, then, may be somewhere between those of *on* and *at* in English.

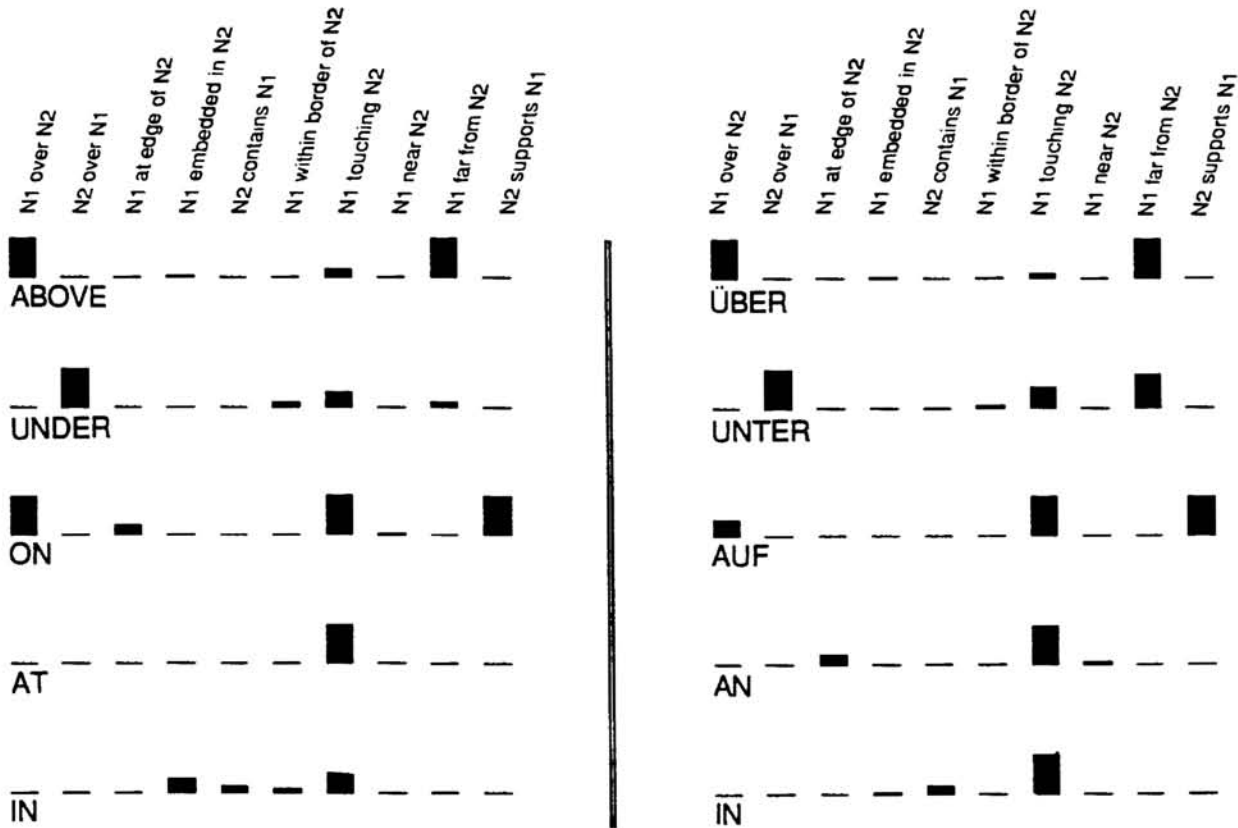

Figure 3: Ideal Meanings of the Prepositions.

## 3.3  TRANSLATION

We made eight translations of the 137-phrase training corpus, four from English to German and four from German to English. The performance for each network over the training corpus is shown in Table 3. The maximum number of phrases translated incorrectly was eight (94.2 percent correct), and the minimum was one wrong (99.3 percent correct). The fact that the English networks learned the training corpus better than the German networks (especially in generating a semantic description for two nouns and a preposition) shows up in the translation task. The English-to-German translations are consistently better than the German-to-English.

Table 3: Number of phrases translated incorrectly (out of 137).

| SIMULATION NUMBER | ENG to GER | GER to ENG |
|---|---|---|
| 1 | 1 | 6 |
| 2 | 3 | 7 |
| 3 | 2 | 6 |
| 4 | 1 | 8 |
| AVG | 1.75 | 6.75 |

# 4   DISCUSSION

Even in this highly constrained and very limited demonstration, the simulations performed using the two databases illustrate how connectionist networks can capture structures in different languages and interact.

The "interlingua" approach to machine translation has not shown promise in practical systems using frameworks based in traditional linguistic theory (Slocum, 1985). The network presented in this paper, however, supports such an approach using a connectionist framework. Of course, even if it is feasible to construct a space with which to represent semantics adequately for the limited domain of concrete uses of locative prepositions, representation of arbitrary semantics is quite another story. On the other hand, semantic representations must be components of any full-scale machine-translation system. In any event, a system that can learn bidirectional mappings between syntax and semantics from a set of examples and extend this learning to novel expressions is a candidate for machine translation (and NLP in general) that warrants further investigation.

We anticipate that any extensive application of back propagation, or any other neural network algorithm, to NLP will involve processing temporal patterns and keeping a dynamic representation of semantic hypotheses, such as the temporal scheme proposed by Elman (1988).

## Acknowledgements

This research was supported in part by NSF grant IRI-8910368 to the first author and by the International Computer Science Institute, which kindly provided the first author with financial support and a stimulating research environment during the summer of 1988.

## References

Cosic, C. and Munro, P. W. (1988) Learning to represent and understand locative prepositional phrases. *10th Ann. Conf. Cognitive Science Society*, 257-262.

Elman, J. L. (1988) Finding structure in time. CRL TR 8801, Center for Research in Language, University of California, San Diego.

Herskovits, Annette (1986) *Language and Spatial Cognition*. Cambridge University Press, Cambridge.

Hinton, Geoffrey (1986) Learning distributed representations of concepts. *8th Ann. Conf. Cognitive Science Society*, 1-12.

Rumelhart, D. E., Hinton, G. and Williams, R. W. (1986) Learning internal representations by error propagation. In: *Parallel Distributed Processing: Explorations in the Microstructure of Cognition*. Vol 1. D. E. Rumelhart and J. L McClelland, eds. Cambridge: MIT/Bradford.

Slocum, J. (1985) A survey of machine translation: its history, current status, and future prospects. *Computational Linguistics, 11*, 1-17.